# Bayesian models for Large-scale Hierarchical Classification

**Siddharth Gopal**
sgopal1@andrew.cmu.edu
Carnegie Mellon University

**Yiming Yang**
yiming@cs.cmu.edu

**Bing Bai**   **Alexandru Niculescu-Mizil**
{bing,alex}@nec-labs.com
NEC Laboratories America, Princeton

## Abstract

A challenging problem in hierarchical classification is to leverage the hierarchical relations among classes for improving classification performance. An even greater challenge is to do so in a manner that is computationally feasible for large scale problems. This paper proposes a set of Bayesian methods to model hierarchical dependencies among class labels using multivariate logistic regression. Specifically, the parent-child relationships are modeled by placing a hierarchical prior over the children nodes centered around the parameters of their parents; thereby encouraging classes nearby in the hierarchy to share similar model parameters. We present variational algorithms for tractable posterior inference in these models, and provide a parallel implementation that can comfortably handle large-scale problems with hundreds of thousands of dimensions and tens of thousands of classes. We run a comparative evaluation on multiple large-scale benchmark datasets that highlights the scalability of our approach and shows improved performance over the other state-of-the-art hierarchical methods.

## 1   Introduction

With the tremendous growth of data, providing a multi-granularity conceptual view using hierarchical classification (HC) has become increasingly important. The large taxonomies for web page categorization at the Yahoo! Directory and the Open Directory Project, and the International Patent Taxonomy are examples of widely used hierarchies. The large hierarchical structures present both challenges and opportunities for statistical classification research. Instead of focusing on individual classes in isolation, we need to address joint training and inference based on the hierarchical dependencies among the classes. Moreover this has to be done in a computationally efficient and scalable manner, as many real world HC problems are characterized by large taxonomies and high dimensionality.

In this paper, we investigate a Bayesian framework for leveraging the hierarchical class structure. The Bayesian framework is a natural fit for this problem as it can seamlessly capture the idea that the models at the lower levels of the hierarchy are specialization of models at the ancestor nodes. We define a hierarchical Bayesian model where the prior distribution for the parameters at a node is a Gaussian centered at the parameters of the parent node. This prior encourages the parameters of nodes that are close in the hierarchy to be similar thereby enabling propagation of information across the hierarchical structure and leading to inductive transfer (sharing statistical strength) among the models corresponding to the different nodes. The strength of the Gaussian prior, and hence the amount of information sharing between nodes, is controlled by its covariance parameter, which is also learned from the data. Modelling the covariance structures gives us the flexibility to incorporate different ways of sharing information in the hierarchy. For example, consider a hierarchical organization of all *animals* with two sub-topics *mammals* and *birds*. By placing feature specific variances, the model can learn that the sub-topic parameters are more similar along common features like 'eyes','claw' and less similar in other sub-topic specific features like 'feathers', 'tail' etc. As another example, the model can incorporate children-specific covariances that allows some sub-topic

parameters to be less similar to their parent and some to be more similar; for e.g. sub-topic *whales* is quite distinct from its parent *mammals* compared to its siblings *felines*, *primates*. Formulating such constraints in non-Bayesian large-margin approaches is not as easy, and to our knowledge has not done before in the context of hierarchical classification. Other advantages of a fully Bayesian treatment are that there no reliance on cross-validation, the outputs have a probabilistic interpretation, and it is easy to incorporate prior domain knowledge.

Our approach shares similarity to the correlated Multinomial logit [18] (corrMNL) in taking a Bayesian approach to model the hierarchical class structure, but improves over it in two significant aspects - scalability and setting hyperparameters. Firstly, CorrMNL uses slower MCMC sampling for inference, making it difficult to scale to problems with more than a few hundred features and a few hundred nodes in the hierarchy. By modelling the problem as a Hierarchical Bayesian Logistic Regression (HBLR), we are able to vastly improve the scalability by 1) developing variational methods for faster inference, 2) introducing even faster algorithms (partial MAP) to approximate the variational inference at an insignificant cost in classification accuracy, and 3) parallelizing the inference. The approximate variational inference (1 plus 2) reduces the computation time by several order of magnitudes (750x) over MCMC, and the parallel implementation in a Hadoop cluster [4] further improves the time almost linearly in the number of processors. These enabled us to comfortably conduct joint posterior inference for hierarchical logistic regression models with tens of thousands of categories and hundreds of thousands of features.

Secondly, a difficulty with the Bayesian approaches, that has been largely side-stepped in [18], is that, when expressed in full generality, they leave many hyperparameters open to subjective input from the user. Typically, these hyper-parameters need to be set carefully as they control the amount of regularization in the model, and traditional techniques such as Empirical Bayes or cross-validation encounter difficulties in achieving this. For instance, Empirical Bayes requires the maximization of marginal likelihood which is difficult to compute in hierarchical logistic models [9] in general, and cross-validation requires reducing the number of free parameters for computational reasons, potentially losing the flexibility to capture the desired phenomena. In contrast, we propose a principled way to set the hyper-parameters directly from data using an approximation to the observed Fisher Information Matrix. Our proposed technique can be easily used to set a large number of hyper-parameters without losing model tractability and flexibility.

To evaluate the proposed techniques we run a comprehensive empirical study on several large scale hierarchical classification problems. The results show that our approach is able to leverage the class hierarchy and obtain a significant performance boost over leading non-Bayesian hierarchical classification methods, as well as consistently outperform flat methods that do not use the hierarchy information.

**Other Related Work:** Most of the previous work in HC has been primarily using large-margin discriminative methods. Some of the early works in HC [10, 14] use the hierarchical structure to decompose the classification problem into sub-problems recursively along the hierarchy and allocate a classifier at each node. The hierarchy is used to partition the training data into node-specific subsets and classifiers at each node are trained independently without using the hierarchy any further. Many approaches have been proposed to better utilize the hierarchical structure. For instance, in [22, 1], the output of the lower-level classifiers was used as additional features for the instance at the top-level classifiers. Smoothing the estimated parameters in naive Bayes classifiers along each path from the root to a leaf node has been tried in [17]. [20, 6] proposed large-margin discriminative methods where the discriminant function at each node takes the contributions from all nodes along the path to the root node, and the parameters are jointly learned to minimize a global loss over the hierarchy. Recently, enforcing orthogonality constraints between parent and children classifiers was shown to achieve state-of-art performance [23].

## 2 The Hierarchical Bayesian Logistic Regression (HBLR) Framework

Define a hierarchy as a set of nodes $Y = \{1, 2...\}$ with the parent relationship $\pi : Y \rightarrow Y$ where $\pi(y)$ is the parent of node $y \in Y$. Let $\mathbf{D} = \{(x_i, t_i)\}_{i=1}^N$ denote the training data where $x_i \in \mathbb{R}^d$ is an instance, $t_i \in T$ is a label, where $T \subset Y$ is the set of leaf nodes in the hierarchy labeled from 1 to $|T|$. We assume that each instance is assigned to one of the leaf nodes in the hierarchy. Let $C_y$ be the set of all children of $y$.

For each node $y \in Y$, we associate a parameter vector $w_y$ which has a Gaussian prior. We set the mean of the prior to the parameter of the parent node, $w_{\pi(y)}$. Different constraints on the covariance matrix of the prior corresponds to different ways of propagating information across the hierarchy. In what follows, we consider three alternate ways to model the covariance matrix which we call M1, M2 and M3 variants of HBLR. In the M1 variant all the siblings share the same spherical covariance matrix. Formally, the generative model for M1 is

$$\mathbf{M1} \quad w_{root} \sim \mathcal{N}(w_0, \Sigma_0), \qquad \alpha_{root} \sim \quad \Gamma(a_0, b_0)$$
$$w_y | \, w_{\pi(y)}, \Sigma_{\pi(y)} \sim \mathcal{N}(w_{\pi(y)}, \Sigma_{\pi(y)}) \quad \forall y, \qquad \alpha_y \sim \quad \Gamma(a_y, b_y) \quad \forall y \notin T$$
$$t \mid x \sim \quad Multinomial(p_1(x), p_2(x), .., p_{|T|}(x)) \quad \forall (x, t) \in \mathbf{D}$$
$$p_i(x) = \quad \exp(w_i^\top x) / \Sigma_{t' \in T} \exp(w_{t'}^\top x) \qquad (1)$$

The parameters of the root node are drawn using user specified parameters $w_0, \Sigma_0, a_0, b_0$. Each non-leaf node $y \notin T$ has its own $\alpha_y$ drawn from a Gamma with the shape and inverse-scale parameters specified by $a_y$ and $b_y$. Each $w_y$ is drawn from the Normal with mean $w_{\pi(y)}$ and covariance matrix $\Sigma_{\pi(y)} = \alpha_{\pi(y)}^{-1} I$. The class-labels are drawn from a Multinomial whose parameters are a soft-max transformation of the $w_y$s from the leaf nodes. This model leverages the class hierarchy information by encouraging the parameters of closely related nodes (parents, children and siblings) to be more similar to each other than those of distant ones in the hierarchy. Moreover, by using different inverse variance parameters $\alpha_y$ for each node, the model has the flexibility to adapt the degree of similarity between the parameters (i.e. parent and children nodes) on a per family basis. For instance it can learn that sibling nodes which are higher in the hierarchy (e.g. *mammals* and *birds*) are generally less similar compared to sibling nodes lower in the hierarchy (e.g. *chimps* and *orangutans*).

Although this model is equivalent to the corrMNL proposed in [18], the hierarchical logistic regression formulation is different from corrMNL and has a distinct advantage that the parameters can be decoupled. As we shall see in Section 3, this enables the use of scalable and parallelizable variational inference algorithms. In contrast, in corrMNL the soft-max parameters are modeled as a sum of contributions along the path from a leaf to the root-node. This introduces two layers of dependencies between the parameters in the corrMNL model (inside the normalization constant as well along the path from leaves to root-node) which makes it less amenable to efficient variational inference. Even if one were to develop a variational approach for the corrMNL parameterization, it would be slower and not efficient for parallelization.

Although the M1 approach is rational, one may argue that it would be beneficial to allow the diagonal elements of the covariance matrix $\Sigma_{\pi(y)}$ to be feature-specific instead of uniform. In our previous example with sub-topics *mammals* and *birds*, we may want $w_{mammals}$, $w_{birds}$ to be commonly close to their parent in some dimensions (e.g., in some common features like 'eyes','breathe' and 'claw') but not in other dimensions (e.g., in *bird* specific features like 'feathers' or 'beak'). We accommodate this by replacing prior $\alpha_y$ using $\alpha_y^{(i)}$ for every feature ($i$). This form of setting the prior is referred to as Automatic Relevant Determination (ARD) and forms the basis of several works such as Sparse Bayesian Learning [19], Relevance Vector Machines [3], etc. For the HC problem, we define the M2 variant of the HBLR approach as:

$$\mathbf{M2} \quad w_y | \, w_{\pi(y)}, \Sigma_{\pi(y)} \sim \mathcal{N}(w_{\pi(y)}, \Sigma_{\pi(y)}) \quad \forall y$$
$$\alpha_y^{(i)} \sim \Gamma(a_y^{(i)}, b_y^{(i)}) \quad i = 1..d, \ \forall y \notin T$$
$$\text{where } \Sigma_{\pi(y)}^{-1} = \text{diag}(\alpha_{\pi(y)}^{(1)}, \alpha_{\pi(y)}^{(2)}, \dots, \alpha_{\pi(y)}^{(d)})$$

Yet another extension of the M1 model would be to allow each node to have its own covariance matrix for the Gaussian prior over $w_y$, not shared with its siblings. This enables the model to learn how much the individual children nodes differ from the parent node. For example, consider topic *mammals* and its two sub-topics *whales* and *carnivores*; the sub-topic *whales* is very distinct from a typical *mammal* and is more of an 'outlier' topic. Such mismatches are very typical in hierarchies; especially in cases where there is not enough training data and an entire subtree of topics is collapsed as a single node. M3 aims to cope up with such differences.

$$\mathbf{M3} \quad w_y | \, w_{\pi(y)}, \Sigma_y \sim \mathcal{N}(w_{\pi(y)}, \Sigma_y) \quad \forall y$$
$$\alpha_y \sim \Gamma(a_y, b_y) \quad \forall y \notin T$$

Note that the only difference between $M3$ and $M1$ is that $M3$ uses $\Sigma_y = \alpha_y^{-1} I$ instead of $\Sigma_{\pi(y)}$ in the prior for $w_y$. In our experiments we found that M3 consistently outperformed the other variants suggesting that such effects are important to model in HC. Although it would be natural to extend

M3 by placing ARD priors instead of the uniform $\alpha_y$, we do not expect to see better performance due to the difficulty in learning a large number of parameters. Preliminary experiments confirmed our suspicions so we did not explore this direction further.

## 3 Inference for HBLR

In this section, we present the inference method for M2 which is harder. The procedure can be easily extended for M1 and M3 [1]. The posterior of M2 is given by

$$p(\mathbf{W}, \boldsymbol{\alpha}|\mathbf{D}) \propto p(\mathbf{D}|\mathbf{W}, \boldsymbol{\alpha})p(\mathbf{W}, \boldsymbol{\alpha}) \tag{2}$$

$$\propto \prod_{(x,t)\in D} \frac{\exp(w_t^\top x)}{\sum_{t'\in T} \exp(w_{t'}^\top x)} \prod_{y\in Y\setminus T} \prod_{i=1}^{d} p(\alpha_y^{(i)}|a_y^{(i)}, b_y^{(i)}) \prod_{y\in Y} p(w_y|w_{\pi(y)}, \Sigma_{\pi(y)})$$

Closed-form solution for the posterior is not possible due to the non-conjugacy between the logistic likelihood and the Gaussian prior, we therefore resort to variational methods to compute the posterior. However, using variational methods are themselves computational intractable in high dimensional scenarios due to the requirement of a matrix inversion which is computationally intensive. Therefore, we explore much faster approximation schemes such as partial MAP inference which are highly scalable. Finally, we show the resulting approximate variational inference procedure can be parallelized in a map-reduce framework to tackle large-scale problems that would be impossible to solve on a single processor.

### 3.1 Variational Inference

Starting with a simple factored form for the posterior, we seek such a distribution $q$ which is closest in *KL* divergence to the true posterior $p$. We use independent Gaussian $q(w_y)$ and Gamma $q(\alpha_y)$ posterior distributions for $w_y$ and $\alpha_y$ per node as the factored representation:

$$q(\mathbf{W}, \boldsymbol{\alpha}) = \prod_{y\in Y\setminus T} q(\alpha_y) \prod_{y\in Y} q(w_y) \propto \prod_{y\in Y\setminus T} \prod_{i=1}^{d} \Gamma(.|\tau_y^{(i)}, \upsilon_y^{(i)}) \prod_{y\in Y} \mathcal{N}(.|\mu_y, \Psi_y)$$

In order to tackle the non-conjugacy inside $p(\mathbf{D}|\mathbf{W}, \boldsymbol{\alpha})$ in (2), we use a suitable lower-bound to the soft-max normalization constant proposed by [5], for any $\beta \in \mathcal{R}$, $\xi_k \in [0, \infty)$

$$\log(\sum_k e^{g_k}) \leq \beta + \sum_k \left[ \frac{g_k - \beta - \xi_k}{2} + \lambda(\xi_k)((g_k - \beta)^2 - \xi_k^2) + \log(1 + e^{\xi_k}) \right]$$

$$\text{where } \lambda(\xi) = \frac{1}{2\xi}\left(\frac{1}{1+e^{-\xi}} - \frac{1}{2}\right)$$

where $\beta$, $\xi_k$ are variational parameters which we can optimize to get the tightest possible bound. For every $(x, y)$ we introduce variational parameters $\beta_x$ and $\xi_{xy}$. We now derive an EM algorithm that computes the posterior in the E-step and maximizes the variational parameters in the M-step.
*Variational E-Step* The local variational parameters are fixed, and the posterior for a parameter is computed by matching the log-likelihood of the posterior with the expectation of log-likelihood under the rest of the parameters. The parameters are updated as[1],

$$\Psi_y^{-1} = I(y \in T) \sum_{(x,t)\in D} 2\lambda(\xi_{xy})xx^\top + \text{diag}(\frac{\tau_{\pi(y)}}{\upsilon_{\pi(y)}}) + |C_y|\,\text{diag}(\frac{\tau_y}{\upsilon_y}) \tag{3}$$

$$\mu_y = \Psi_y \left( I(y \in T) \sum_{(x,t)\in D} (I(t=y) - \frac{1}{2} + 2\lambda(\xi_{xy})\beta_x)x + \text{diag}(\frac{\tau_{\pi(y)}}{\upsilon_{\pi(y)}})\mu_{\pi(y)} + \text{diag}(\frac{\tau_y}{\upsilon_y})\sum_{c\in C_y}\mu_c \right)$$

$$\upsilon_y^{(i)} = b_y^{(i)} + \sum_{c\in C_y} \Psi_y^{(i,i)} + \Psi_c^{(i,i)} + (\mu_y^{(i)} - \mu_c^{(i)})^2 \qquad \text{and } \tau_y^{(i)} = a_y^{(i)} + \frac{|C_y|}{2} \tag{4}$$

*Variational M-Step* We keep the parameters of the posterior distribution fixed and maximize the variational parameters $\xi_{xy}, \beta_x$. Refer to [5] for detailed M-step derivations,

$$\xi_{xy}^2 = x^\top \text{diag}(\frac{\tau_y}{\upsilon_y})x + (\beta_x - \mu_y^\top x)^2 \qquad \beta_x = (.5(.5|T| - 1) + \sum_{y\in T} \lambda(\xi_{xy})\mu_y^\top x) / \sum_{y\in T} \lambda(\xi_{xy})$$

*Class-label Prediction* After computing the posterior, one way to compute the probability of a target class-label given a test instance is to simply plugin the posterior mean for prediction. A more principled way would be to compute the predictive distribution of the target class label *l* given the

test instance,

$$p(l|x) = \int p(l, \mathbf{W}|x)d\mathbf{W} \approx \int p(l|\mathbf{W}, x)q(\mathbf{W})d\mathbf{W} \tag{5}$$

The above integral cannot be computed in closed form and people have often resorted to probit approximations [16]. We take an alternative route by calculating the joint posterior $p(l, \mathbf{W}|x)$ by variational approximations. We assume the following factored form for the predictive distribution,

$$\tilde{q}(l, \mathbf{W}) = \prod_{y \in T} \tilde{q}(w_y)\tilde{q}(l_y) \equiv \prod_{y \in T} \mathcal{N}(.|\tilde{\mu}_y, \check{\Psi}_y)Bern(.|\tilde{p}_y)$$

The posterior can be calculated as before, by introducing variational parameters $\tilde{\xi}_{xy}$, $\tilde{\beta}_x$ and matching the log likelihoods. Substituting $\tilde{q}(l, \mathbf{W})$ in (5), we see that the predictive distribution is given by $\tilde{q}(l)$ and the target class label is given by $\arg\max_{y \in T} \tilde{p}_y$.

## 3.2 Partial MAP Inference

In most applications, the requirement for a matrix inversion in step (3) could be demanding. In such scenarios, we split the inference into two stages, first calculating the posterior of $w_y$ using MAP solution, and second calculating the posterior of $\alpha_y$. In the first stage, we find the MAP estimate $w_y^{map}$ and then use laplace approximation to approximate the posterior using a separate Normal distribution for each dimension, thereby leading to a diagonal covariance matrix. Note that due to the laplace approximation, $w_y^{map}$ and the posterior mean $\mu_y$ coincide.

$$\boldsymbol{\mu} = w_y^{map} = \arg\max_{\mathbf{W}} \sum_{y \in T} -\frac{1}{2}(w_y - w_{\pi(y)})^\top \operatorname{diag}(\frac{\tau_{\pi(y)}}{v_{\pi(y)}})(w_y - w_{\pi(y)}) + \log p(\mathbf{D}|\mathbf{W}, \boldsymbol{\alpha}) \tag{6}$$

$$(\Psi_y^{(i,i)})^{-1} = \sum_{(x,t) \in D_y} x^{(i)} p_{xy}(1 - p_{xy})x^{(i)}$$

where $p_{xy}$ is the probability that training instance $x$ is labeled as $y$. The $\arg\max$ in (6) can be computed for all $\mu_y$ at the same time using optimization techniques like LBFGS [13]. For the second stage, parameters $\tau_y$ and $v_y$ are updated using (4). Full MAP inference is also possible by performing an alternating maximization between $w_y, \alpha_y$ but we do not recommend it as there is no gain in scalability compared to partial MAP Inference and it loses the posterior distribution of $\alpha_y$.

## 3.3 Parallelization

For large hierarchies, it might be impractical to learn the parameters of all classes, or even store them in memory, on a single machine. We therefore, devise a parallel memory-efficient implementation scheme for our partial MAP Inference. There are 4 sets of parameters that are updated - $\{\mu_y, \Psi_y, \tau_y, \nu_y\}$. The $\Psi_y, \tau_y, \nu_y$ can be updated in parallel for each node using (3),(4).

For $\mu$, the optimization step in (6) is not easy to parallelize since the $w$'s are coupled together inside the soft-max function. To make it parallelizable we replace the soft-max function in (1) with multiple binary logistic functions (one for each terminal node), which removes the coupling of parameters inside the log-normalization constant. The optimization can now be done in parallel by making the following observations - firstly note that the optimization problem in (6) is concave maximation, therefore any order of updating the variables reaches the same unique maximum. Secondly, note that the interactions between the $w_y$'s are only through the parent and child nodes. By fixing the parameters of the parent and children, the parameter $w_y$ of a node can be optimized independently of the rest of the hierarchy. One simple way to parallelize is to traverse the hierarchy level by level, optimize the parameters at each level in parallel, and iterate until convergence. A better way that achieves a larger degree of parallelization is to iteratively optimize the odd and even levels - if we fix the parameters at the odd levels, the parameters of parents and the children of all nodes at even levels are fixed, and the $w_y$'s at all even levels can be optimized in parallel. The same goes for optimizing the odd level parameters. To aid convergence we interleave the $\mu, \Psi$ updates with the $\tau, \nu$ updates and warm-start with the previous value of $\mu_y$. In practice, for the larger hierarchies we observed speedups linear in the number of processors. Note that the convergence follows from viewing this procedure as block co-ordinate ascent on a concave differentiable function [15].

We tested our parallelization framework on a cluster running map-reduce based Hadoop 20.2 with 64 worker nodes with 8 cores and 16GB RAM each. We used Accumulo 1.4 key-value store for fast retrieve-update of the $w_y$s. On this hardware, our experiments on the largest dataset with 15358 class labels and 347256 features took just 38 minutes. Although the map-reduce framework is not a requirement; it is a ubiquitous paradigm in distributed computing and having an implementation compatible with it is a definite advantage.

Table 1: Dataset Statistics

| Dataset | #Training | #Testing | #Class-Labels | #Leaf-labels | Depth | #Features |
|---|---|---|---|---|---|---|
| **CLEF** | 10000 | 1006 | 87 | 63 | 4 | 89 |
| **NEWS20** | 11260 | 7505 | 27 | 20 | 3 | 53975 |
| **LSHTC-small** | 4463 | 1858 | 1563 | 1139 | 6 | 51033 |
| **LSHTC-large** | 93805 | 34905 | 15358 | 12294 | 6 | 347256 |
| **IPC** | 46324 | 28926 | 552 | 451 | 4 | 541869 |

## 4 Setting prior parameters

The $w_0, \Sigma_0$ represent the overall mean and covariance structure for the $w_y$. We set $w_0 = 0$ and $\Sigma_0 = I$ because of their minimal effect on the rest of the parameters. The $a_y^{(i)}, b_y^{(i)}$ are variance components such that $\frac{b_y^{(i)}}{a_y^{(i)}}$ represents the expected variance of the $w_y^{(i)}$. Typically, choosing these parameters is difficult before seeing the data. The traditional way to overcome this is to learn $\{a_y, b_y\}$ from the data using Empirical Bayes. Unfortunately, in our proposed model, one cannot do this as each $\{a_y, b_y\}$ is associated with a single $\alpha_y$. Generally, we need more than one sample value to learn the prior parameters effectively [7].

We therefore resort to a data dependent way of setting these parameters by using an approximation to the observed Fisher Information matrix. We first derive on a simpler model and then extend it to a hierarchy. Consider the following binary logistic model with unknown $w$ and let the Fisher Information matrix be $I$ and observed Fisher Information $\hat{I}$

$$Y \mid x \sim Bernoulli(\frac{\exp(w^\top x)}{1 + \exp(w^\top x)}); \qquad I = E\left[p(x)(1 - p(x))xx^\top\right], \hat{I} = \sum_{(x,t) \in D} \hat{p}(x)(1 - \hat{p}(x))xx^\top$$

It is well known that $I^{-1}$ is the asymptotic covariance of the MLE estimator of $w$, so reasonable guess for the covariance of a Gaussian prior over $w$ could be the observed $\hat{I}^{-1}$ from a dataset $D$. The problem with $\hat{I}^{-1}$ is that we do not have a good estimate $\hat{p}(x)$ for a given $x$ as we have exactly one sample for a given $x$ i.e each instance $x$ is labeled exactly once with certainty, therefore $\hat{p}(x)(1 - \hat{p}(x))$ will always be zero. Therefore we approximate $\hat{p}(x)$ as the sample prior probability independent of $x$, i.e. $\hat{p}(x) = \hat{p} = \Sigma_{(x,t) \in D} \frac{t}{|D|}$. Now, the prior on the covariance of $w_y$ can be set such that the expected covariance is $\hat{I}^{-1}$. To extend this to HC, we need to handle multiple classes, which can be done by estimating $\hat{I}(y)^{-1}$ for each $y \in T$, as well handle multiple levels, which can be done by recursively setting $a_y, b_y$ as follows,

$$(a_y^{(i)}, b_y^{(i)}) = \begin{cases} (\sum_{c \in C_y} a_c^{(i)}, \sum_{c \in C_y} b_c^{(i)}) & \text{if } y \notin T \\ (1, \hat{I}(y)^{-1(i,i)}) & \text{if } y \in T \end{cases}$$

where $\hat{I}(y)$ is the observed Fisher Information matrix for class label $y$. This way of setting the priors is similar to the method proposed in [12], the key differences are in approximating $p(x)(1 - p(x))$ from the data rather using $p(x) = \frac{1}{2}$, extension to handle multiple classes as well as hierarchies.

We also tried other popular strategies such as setting improper gamma priors $\Gamma(\epsilon, \epsilon)$ $\epsilon \to 0$ widely used in many ARD works (which is equivalent to using type-2 ML for the $\alpha$'s if one uses variational methods [2]) and Empirical Bayes using a single $a$ and $b$ (as well as other Empirical Bayes variants). Neither of worked well, the former being to be too sensitive to the value of $\epsilon$ which is in agreement with the observations made by [11] and the latter constraining the model by using a single $a$ and $b$. We do not discuss this any further due to lack of space.

## 5 Experiments Results

Throughout our experiements, we used 4 popular benchmark datasets (Table 1) with the recommended train-test splits - CLEF[8], NEWS20[2], LSHTC-{small,large}[3], IPC[4].

First, to evaluate the speed advantage of the variational inference, we compare the full variational {M1,M2,M3}-var and partial MAP {M1,M2,M3-map} inference [5] for the three variants of HBLR to the MCMC sampling based inference of CorrMNL [18]. For CorrMNL, we used the implementation as provided by the authors[6]. We performed sampling for 2500 iterations with 1000 for burn-in.

Table 2: *Comparison with CorrMNL:* Macro-$F_1$ and Micro-$F_1$ on the **CLEF** dataset

| | CorrMNL | {M1,M2,M3}-var | | | {M1,M2,M3}-map | | | {M1,M2,M3}-flat | | |
|---|---|---|---|---|---|---|---|---|---|---|
| | | M1 | M2 | M3 | M1 | M2 | M3 | M1 | M2 | M3 |
| *Macro-f1* | 55.59 | 56.67 | 51.23 | **59.67** | 55.53 | 54.76 | 59.65 | 52.13 | 48.78 | 55.23 |
| *Micro-f1* | 81.10 | 81.21 | 79.92 | **81.61** | 80.88 | 80.25 | 81.41 | 79.82 | 77.83 | 80.52 |
| *Time (mins)* | 2279 | 79 | 81 | 80 | 3 | 3 | 3 | 3 | 3 | 3 |

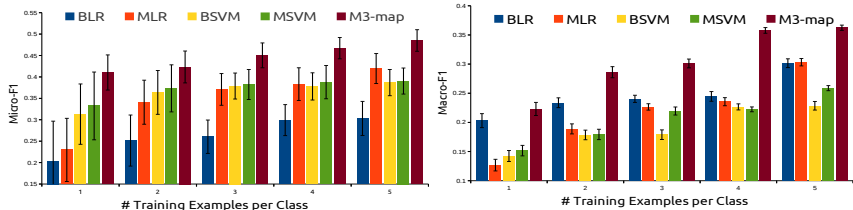

Figure 1: Micro-$F_1$ (left) & Macro-$F_1$ (right) on the CLEF dataset with limited number of training examples.

Re-starts with different initialization values gave the same results for both MCMC and variational methods. All models were run on a single CPU without parallelization. We used the small CLEF[8] dataset in order to be able to run CorrMNL model in reasonable time. The results are presented in Table 2. For an informative comparison, we also included the results of {M1,M2,M3}-flat, our proposed approach using a flat hierarchy. With regards to scalability, partial MAP inference is the most scalable method being orders of magnitude faster (750x) than CorrMNL. Full variational inference, although less scalable as it requires $O(d^3)$ matrix inversions in the feature space, is still orders of magnitude faster (20x) than CorrMNL. In terms of performance, we see that the partial MAP inference for the HBLR has only small loss in performance compared to the full variational inference while having similar training time to the flat approach that does not model the hierarchy ({M1,M2,M3}-flat).

Next, we compare the performance of HBLR to several other competing approaches:

1. **Hierarchical Baselines**: We selected 3 representative hierarchical methods that have shown to have state-of-the-art performance - Hierarchical SVM [6] (HSVM), a large-margin discriminative method with path-dependent discriminant function. Orthogonal Transfer [23] (OT), a method enforcing orthogonality constraints between the parent node and children and Top-down Classification [14] (TD) Top-down decision making using binary SVMs trained at each node.

2. **Flat Baselines**: Typical flat approaches which do not make use of the hierarchy. We tested One-versus rest Binary logistic Regressions (BLR), Multiclass Logistic Regression (MLR), One-versus Rest Binary SVMs (BSVM), and Multiclass SVM (MSVM) [21].

For all competing approaches, we tune the regularization parameter using 5 fold CV with a range of values from $10^{-5}$ to $10^5$. For the HBLR models, we used partial MAP Inference because full variational is not scalable to high dimensions. The IPC and LSHTC-large are very large datasets so we are unable to test any method other than our parallel implementation of HBLR, and BLR, BSVM which can be trivially parallelized. Although TD can be parallelized we did not pursue this since TD did not achieve competitive performance on the other datasets. Parallelizing the other methods is not obvious and has not been discussed in previous literature to the best of our knowledge.

Table 3 summarizes the results obtain by the different methods. The performance was measured using the standard macro-$F_1$ and micro-$F_1$ measures [14]. The significance tests are performed using sign-test for Micro-$F_1$ and a wilcoxon rank test on the Macro-$F_1$ scores. For every data collection, each method is compared to the best performing method on that dataset. The null hypothesis is that there is no significanct difference between the two systems being compared, the alternative is that the best-performing-method is better. Among M1,M2 and M3, the performance of M3 seems to be consistently better than M1, followed by M2. Although M2 is more expressive than M1, the benefit of a better model seems to be offset by the difficulty in learning a large number of parameters.

Comparing to the other hierarchical baselines, M3 achieves significantly higher performance on all datasets, showing that the Bayesian approach is able to leverage the information provided in the class hierarchy. Among the baselines, we find that the average performance of HSVM is higher than the TD, OT. This can be partially explained by noting that both OT and TD are greedy top-down classification methods and any error made in the top level classifications propagates down to

Table 3: Macro-$F_1$ and Micro-$F_1$ on the 4 datasets. Bold faced number indicate best performing method. The results of the significance tests are denoted * for a p-value less than 5% and $^\dagger$ for p-value less than 1%.

| | {M1,M2,M3}-map | | | Hierarchical methods | | | Flat methods | | | |
|---|---|---|---|---|---|---|---|---|---|---|
| | M1 | M2 | M3 | HSVM | OT | TD | BLR | MLR | BSVM | MSVM |
| **CLEF** | | | | | | | | | | |
| *Macro-f1* | 55.53$^\dagger$ | 54.76$^\dagger$ | **59.65** | 57.23* | 37.12$^\dagger$ | 32.32$^\dagger$ | 53.26$^\dagger$ | 54.76$^\dagger$ | 48.59$^\dagger$ | 54.33$^\dagger$ |
| *Micro-f1* | 80.88* | 80.25* | **81.41** | 79.72$^\dagger$ | 73.84$^\dagger$ | 70.11$^\dagger$ | 79.92$^\dagger$ | 80.52$^\dagger$ | 77.53$^\dagger$ | 80.02$^\dagger$ |
| **NEWS20** | | | | | | | | | | |
| *Macro-f1* | 81.54 | 80.91* | 81.69 | 80.04$^\dagger$ | 81.20 | 80.86* | 82.17 | 81.82 | **82.32** | 81.73 |
| *Micro-f1* | 82.24* | 81.54* | 82.56* | 80.79* | 81.98* | 81.20$^\dagger$ | 82.97 | 82.56* | **83.10** | 82.47* |
| **LSHTC-small** | | | | | | | | | | |
| *Macro-f1* | 28.81$^\dagger$ | 25.81$^\dagger$ | **30.81** | 21.95$^\dagger$ | 19.45$^\dagger$ | 20.01$^\dagger$ | 28.12$^\dagger$ | 28.38* | 28.62* | 28.34* |
| *Micro-f1* | 45.48 | 43.31$^\dagger$ | **46.03** | 39.66$^\dagger$ | 37.12$^\dagger$ | 38.48$^\dagger$ | 44.94$^\dagger$ | 45.20 | 45.21* | 45.62 |
| **LSHTC-large** | | | | | | | | | | |
| *Macro-f1* | 28.32* | 24.93$^\dagger$ | **28.76** | - | - | - | 27.91* | - | 27.89* | - |
| *Micro-f1* | 43.98 | 43.11$^\dagger$ | **44.05** | - | - | - | 43.98 | - | 44.03 | - |
| **IPC** | | | | | | | | | | |
| *Macro-f1* | 50.43$^\dagger$ | 47.45$^\dagger$ | **51.06** | - | - | - | 48.29$^\dagger$ | - | 45.71$^\dagger$ | - |
| *Micro-f1* | 55.80* | 54.22$^\dagger$ | **56.02** | - | - | - | 55.03$^\dagger$ | - | 53.12$^\dagger$ | - |

the leaf node; in contrast to HSVM which uses an exhaustive search over all labels. However, the result of OT do not seem to support the conclusions in [23]. We hypothesize two reasons - firstly, the orthogonality condition which is assumed in OT does not hold in general, secondly, unlike [23] we use cross-validation to set the underlying regularization parameters rather than setting them arbitrarily to 1 (which was used in [23]).

Surprisingly, the hierarchical baselines (HSVM,TD and OT) experience a very large drop in performance on LSHTC-small when compared to the flat baselines, indicating that the hierarchy information actually mislead these methods rather than helping them. In contrast, M3 is consistently better than the flat baselines on all datasets except NEWS20. In particular, M3 performs significantly better on the largest datasets, especially in Macro-$F_1$, showing that even very large class hierarchies can convey very useful information, and highlighting the importance of having a scalable, parallelizable hierarchical classification algorithm.

To further establish the importance of modeling the hierarchy, we test our approach under scenarios when the number of training examples is limited. We expect the hierarchy to be most useful in such cases as it enables of sharing of information between class parameters. To verify this, we progressively increased the number of training examples per class-label on the CLEF dataset and compared M3-map with the other best performing methods. Figure 1 reports the results of M3-map, MLR, BSVM, MSVM averaged over 20 runs. The results shows that M3-map is significantly better than the other methods especially when the number of examples is small. For instance, when there is exactly one training example per class, M3-map achieves a whopping 10% higher Micro-$F_1$ and a 2% higher Macro-$F_1$ than the next best method. We repeated the same experiments on the NEWS20 dataset but however did not find an improved performance even with limited training examples suggesting that the hierarchical methods are not able to leverage the hierarchical structure of NEWS20.

## 6    Conclusion

In this paper, we presented the HBLR approach to hierarchical classification, focusing on scalable ways to leverage hierarchical dependencies among classes in a joint framework. Using a Gaussian prior with informative mean and covariance matrices, along with fast variational methods, and a practical way to set hyperparameters, HBLR significantly outperformed other popular HC methods on multiple benchmark datasets. We hope this study provides useful insights into how hierarchical relationships can be successfully leveraged in large-scale HC. In future, we would like to adapt this approach to equivalent non-bayesian large-margin discriminative counterparts.

**ACKNOWLDEGMENTS:** This work is supported, in part, by the NEC Laboratories America, Princeton under 'NEC Labs Data Management University Awards' and the National Science Foundation (NSF) under grant IIS_1216282. A major part of work was accomplished while the first author was interning at NEC Labs, Princeton.

## Footnotes

[1] Complete derivations are presented in the extended version located at http://www.cs.cmu.edu/~sgopal1.

[2] http://people.csail.mit.edu/jrennie/20Newsgroups/     [3] http://lshtc.iit.demokritos.gr/

[4] http://www.wipo.int/classifications /ipc/en/support/     [5] Code available at http://www.cs.cmu.edu/˜sgopal1

[6] http://www.ics.uci.edu/ babaks/Site/Codes.html

# References

[1] P.N. Bennett and N. Nguyen. Refined experts: improving classification in large taxonomies. In *SIGIR*, 2009.

[2] C.M. Bishop. *Pattern recognition and machine learning*.

[3] C.M. Bishop and M.E. Tipping. Bayesian regression and classification. 2003.

[4] D. Borthakur. The hadoop distributed file system: Architecture and design. *Hadoop Project Website*, 11:21, 2007.

[5] G. Bouchard. Efficient bounds for the softmax function. 2007.

[6] L. Cai and T. Hofmann. Hierarchical document categorization with support vector machines. In *CIKM*, pages 78–87. ACM, 2004.

[7] George Casella. Empirical bayes method - a tutorial. Technical report.

[8] I. Dimitrovski, D. Kocev, L. Suzana, and S. Džeroski. Hierchical annotation of medical images. In *IMIS*, 2008.

[9] C.B. Do, C.S. Foo, and A.Y. Ng. Efficient multiple hyperparameter learning for log-linear models. In *Neural Information Processing Systems*, volume 21, 2007.

[10] S. Dumais and H. Chen. Hierarchical classification of web content. In *SIGIR*, 2000.

[11] A. Gelman. Prior distributions for variance parameters in hierarchical models. *BA*.

[12] R.E. Kass and R. Natarajan. A default conjugate prior for variance components in generalized linear mixed models. *Bayesian Analysis*, 2006.

[13] D.C. Liu and J. Nocedal. On the limited memory bfgs method for large scale optimization. *Mathematical programming*, 45(1):503–528, 1989.

[14] T.Y. Liu, Y. Yang, H. Wan, H.J. Zeng, Z. Chen, and W.Y. Ma. Support vector machines classification with a very large-scale taxonomy. *ACM SIGKDD*, pages 36–43, 2005.

[15] Z.Q. Luo and P. Tseng. On the convergence of the coordinate descent method for convex differentiable minimization. *Journal of Optimization Theory and Applications*, 72(1):7–35, 1992.

[16] D.J.C. MacKay. The evidence framework applied to classification networks. *Neural computation*, 1992.

[17] A. McCallum, R. Rosenfeld, T. Mitchell, and A.Y. Ng. Improving text classification by shrinkage in a hierarchy of classes. In *ICML*, pages 359–367, 1998.

[18] B. Shahbaba and R.M. Neal. Improving classification when a class hierarchy is available using a hierarchy-based prior. *Bayesian Analysis*, 2(1):221–238, 2007.

[19] M.E. Tipping. Sparse bayesian learning and the relevance vector machine. *JMLR*, 1:211–244, 2001.

[20] I. Tsochantaridis, T. Joachims, T. Hofmann, and Y. Altun. Large margin methods for structured and interdependent output variables. *JMLR*, 6(2):1453, 2006.

[21] J. Weston and C. Watkins. Multi-class support vector machines. Technical report, 1998.

[22] G.R. Xue, D. Xing, Q. Yang, and Y. Yu. Deep classification in large-scale text hierarchies. In *SIGIR*, pages 619–626. ACM, 2008.

[23] D. Zhou, L. Xiao, and M. Wu. Hierarchical classification via orthogonal transfer. Technical report, MSR-TR-2011-54, 2011.

